# Pseudo-Siamese Blind-Spot Transformers for Self-Supervised Real-World Denoising

**Yuhui Quan**\*, **Tianxiang Zheng**\*
School of Computer Science and Engineering
South China University of Technology

**Hui Ji**\*
Department of Mathematics
National University of Singapore

## Abstract

Real-world image denoising remains a challenge task. This paper studies self-supervised image denoising, requiring only noisy images captured in a single shot. We revamping the blind-spot technique by leveraging the transformer's capability for long-range pixel interactions, which is crucial for effectively removing noise dependence in relating pixel–a requirement for achieving great performance for the blind-spot technique. The proposed method integrates these elements with two key innovations: a directional self-attention (DSA) module using a half-plane grid for self-attention, creating a sophisticated blind-spot structure, and a Siamese architecture with mutual learning to mitigate the performance impacts from the restricted attention grid in DSA. Experiments on benchmark datasets demonstrate that our method outperforms existing self-supervised and clean-image-free methods. This combination of blind-spot and transformer techniques provides a natural synergy for tackling real-world image denoising challenges.

## 1 Introduction

Images taken with digital cameras inevitably acquire noise from various sources. Image denoising aims to recover a clean (noise-free) image from its noisy counterpart, serving as an important technique in many low-level vision tasks. Recently, deep learning has prominently driven advancements in image denoising; see *e.g.*, [1, 2, 3, 4, 5, 6, 7, 8, 9, 10, 11]. Early works typically take a supervised learning approach to train a neural network (NN) on a dataset with paired noisy and clean images, where noisy images are synthesized by corrupting clean images with Additive White Gaussian Noise (AWGN). As observed in [6, 12], a denoising NN trained on AWGN fails to generalize well on real-world noisy images due to the statistical distribution gap between AWGN and real-world noise.

To address this issue in supervised learning, several studies have concentrated on constructing datasets with paired real-world noisy and clean images, such as the Smartphone Image Denoising Dataset (SIDD) [13] and the Darmstadt Noise Dataset (DND) [14]. However, creating these datasets is labor-intensive, involving multiple acquisitions of images of the same scene, requiring rigorous image alignment, and is inapplicable to dynamic scenes. Additionally, the statistical properties of real noise vary for different camera systems and settings. These limitations restrict the wider application of supervised-learning-based solutions in real-world image denoising.

To address the difficulties of creating paired noisy-clean image datasets that capture the noise characteristics of the targeted testing data, some approaches train denoising NNs using unpaired noisy and clean images [15, 16, 17, 8]. Other studies [18, 19, 20] use multiple noisy images of the same scene as training data. While these methods relax data collection requirements, limitations remain. The former still requires clean images, which is challenging to collect, especially in scientific/medical imaging. The latter involves rigorous image alignment, making it unsuitable for dynamic scenes.

**Self-supervised denoising methods:** In recent years, there has been increasing interest in self-supervised denoising methods, where network training requires only a set of *noisy images captured in a single shot*. Among these methods, Blind-Spot Networks (BSNs) and their variations are particularly popular; see, *e.g.* [21, 22, 23, 24, 25, 26, 27, 28, 29, 30]. The concept of BSNs has played a significant role in self-supervised learning for various denoising-related tasks. In principle, through specific architectural design, a BSN estimates each output pixel from the surrounding noisy pixels, excluding the corresponding one. This design prevents the network from converging to an identity mapping when trained to minimize the distance between the output of the NN and the input noisy image, as no clean image is available. Note that pixel-wise noise independence is critical for BSN to work effectively.

Recently, transformers, NNs utilizing self-attention (SA) for sequence modeling, have shown great performance in many applications, including image denoising; see, for example, [31, 32, 33, 34, 35]. Compared to Convolutional Neural Networks (CNNs), which use multiple convolution layers to extend the receptive field and connect distant pixels, transformers directly model interactions between distant pixels via attention, capturing long-range dependencies more effectively.

The effectiveness of transformers or SA in connecting distant pixels is particularly attractive for self-supervised denoising, especially for Blind-Spot Networks (BSNs). The ability of transformers to exploit long-range dependencies aligns well with the need for pixel-wise noise independence in BSNs to achieve optimal performance, as noise on distant pixels is more likely to be independent than on neighboring pixels. While the integration of transformers with BSNs is very promising, it has not been well studied in self-supervised denoising.

Currently, there are few studies [28, 29, 36] integrate SA and BSN for self-supervised denoising. LG-BSN [28] used channel SA [34] with dilated convolutions for blind-spot learning but did not utilize spatial SA. SS-BSN [29] integrated a lightweight spatial SA module into a BSN with dilated convolutions but lacked rigorous justification of blind-spot constraints. SwinIA [36] applied masking to the Swin Transformer [31] attention matrix, targeting only AWGN. As a result, these transformer-based methods do not outperform recent CNN-based approaches [30] in denoising real-world images. It remains an open question how to fully leverage the effectiveness of long-range dependence in transformers for better performance in self-supervised real-world denoising.

**Main idea of our approach:** The direct combination of SA and BSD, as done in [29], may compromise the blind-spot property [28]. Another approach is to mask out input pixels, as in [21, 22]. However, pixel masking can reduce the integrity and the accuracy of long-range feature interactions within transformer blocks. It also limits the number of available pixels for loss calculation, thereby leads to sub-optimal performance. We address these issues with SelfFormer, a self-supervised transformer model featuring a built-in blind-spot structure that avoids input pixel masking.

Our SelfFormer is built on a directional self-attention (DSA) mechanism, inspired by directional convolution [23]. Unlike plain self-attention, where the attention window spreads out in all directions (see Figure1(a)), DSA's attention window is constrained to a half-plane excluding the token itself (see Figure 1(b)). This unidirectional flow ensures that information broadcasted by a token does not return to it, even after multiple DSA applications, thus creating a blind-spot mechanism.

SelfFormer has four branches, each using DSA in one direction: left, right, up, and down. Combining outputs from all branches allows attention to extend arbitrarily far in every direction without including the center pixel. To improve computational efficiency, we use a gridding scheme on the directional attention window. In addition, we introduce a channel attention (CA) mechanism with blind-spot properties to capture channel interdependence, complementing the spatial similarity captured by SA.

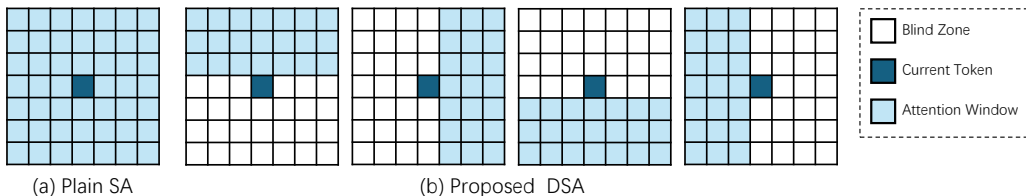

(a) Plain SA          (b) Proposed  DSA

Figure 1: Illustration of basic idea of our approach.

To further enhance the effectiveness of DSA caused by its restricted attention locations, we introduce a pseudo-Siamese architecture for SelfFormer. One sub-NN, *SelfFormer-D*, employs DSA with four branches, while the other, *SelfFormer-F*, uses full-grid SA with only one branch. SelfFormer-D's four branches share weights and process four rotated inputs, allowing identical learnable weights for DSA and full-grid SA in the pseudo-Siamese learning. Despite different structures, this setup ensures both sub-networks have consistent weights. SelfFormer-D and SelfFormer-F, with their different inductive biases, provide mutual regularization through joint learning. Due to its better capability of exploiting long-range dependencies with unconstrained attention windows, SelfFormer-F is used for inference.

**Contributions:** The main contributions of this paper are summarized as follows:

- SelfFormer, an efficient self-supervised transformer, is introduced for real-world image denoising, integrating the blind-spot mechanism with transformers.
- DSA, a specific type of SA, is developed to embed the blind-spot mechanism in transformers.
- A pseudo-Siamese architecture is introduced to address the potential negative performance impact of the restricted attention locations caused by the blind-spot mechanism.

The proposed SelfFormer for denoising real-world images is evaluated on two popular datasets, SIDD and DND, and compared with many existing image denoisers. The results showed that SelfFormer outperforms existing self-supervised denoisers and other methods that not using clean images.

## 2 Related Work

**Supervised denoising:** The study of supervised denoisers focus on designed network architecture. Most existing methods are based on CNNs; see *e.g.* [1, 2, 3, 4, 5, 6, 7, 8, 9, 10, 37, 11]. Recently, transformers have become a primary choice due to their performance advantage [31, 32, 33, 34, 35]. Training data is another concern. Instead of using AWGN for data synthesis, CBDNet [12] synthesized heteroscedastic Gaussian noise and processed it through a Image Signal Processor. Zhou *et al.* [38] trained a noise estimator and a denoiser with mixed AWGN and random-valued impulse noise and then utilized pixel-shuffle down-sampling to adapt the trained model to real noises. For better generalization and evaluation, two widely-used real-world noisy datasets were constructed in [13, 14], respectively. However, large-scale paired data collection remains a challenge.

**Weakly-supervised denoising:** The related approaches can be categorized into learning on unpaired image data, and learning on paired noisy image data. The former leverages unpaired noisy and clean images, using either generative adversarial networks [15, 16] or flow-based methods [17, 20]. The latter use multiple noisy images of the same scene to train a denoiser [18, 19, 20]. Note that such multi-capture of noisy images still requires image alignment and is not applicable to dynamic scenes.

**Self-supervised denoising:** Self-supervised denoisers are trained using only a set of noisy images captured in a single shot. BSN is one popular self-supervised denoiser. Masking-based BSNs (e.g., Noise2Void [21] and Noise2Self [22]) address overfitting caused by the absence of clean images in the loss function by masking a portion of input pixels and predicting them using the remaining pixels. Noise2Same [24] introduces an additional self-reconstruction loss to utilize center pixels' information. Blind2Unblind [25] trains a masker to better preserve valuable pixels in the masked input. SASL [39] gives separate treatment to flat and textured regions in masking-based self-supervision.

A large percentage of input pixels are not used in the loss function when they are masked out, leading to sub-optimal performance [23]. To address this, some works design specific network architectures to ensure the receptive-field center is not seen by the corresponding pixel during prediction. Laine19 *et al.* [23] occludes half of the receptive fields of a CNN in four different directions. D-BSN [26] applies a center-masked convolution, followed by a series of dilated convolutions with specific step sizes. MM-BSN [40] uses multiple convolutional kernels masked in different shapes.

The less correlated the noise of related pixels is, the better the denoiser performs. However, in real images, neighboring pixels' noise is highly correlated. One solution is to relate distant pixels. AP-BSN [27] uses pixel-shuffle downsampling with high strides in training, and low strides in testing. PUCA [30] leverages patch-unshuffle/shuffle to expand receptive fields for relating distant pixels. LG-BPN [28] uses channel SA with dilated convolution for relating distant pixels and densely-sampled patch-masked convolution to recover local structures. SS-BSN [29] combines grid SA [41]

with a simplified D-BSN [26]. SwinIA [36] masks the diagonal of the attention matrix in a Swin Transformer [31].

Besides BSNs, there are also other approaches which are architecture-independent. SURE-based methods (*e.g.* [42]), introduce Stein's unbiased estimator [43] to regularize the training. Recorruption-based methods, *e.g.*, Noisier2Noise [44], R2R [45], IDR [46], and Zheng *et al.* [47], define the loss using pairs derived from the input noisy image to simulate a supervised loss. Sampling-based methods (*e.g.* Neighbor2Neighbor [48]) form training pairs using a random neighbor sampler on the noisy image. Score-based methods, *e.g.*, Noise2Score [49], NDASSID [50], and Xie *et al.* [51], perform self-supervised training via score matching [52]. Disentanglement-based methods, *e.g.*, CVF-SID [53] use a cyclic loss function to decompose the noisy image into clean and noisy components.

**Zero-shot denoising:** This approach performs per-sample self-supervised training. The sparse-coding-based denoisers learn a dictionary [54, 55, 56, 57] from the noisy image, and the denoising output is defined as a sparse approximation to the input based on the learned dictionary. The NN-based denoiser, such as DIP [58], Self2Self[59], NoisyAsClean[60], Noise2Fast [61] and ScoreDVI [62], also follow this paradigm. However, this approach is computationally costly when applied to a large number of images.

## 3 Methodology

In this section, we give a detailed description of our proposed SelfFormer, which effectively integrates the blind-spot mechanism with transformers for self-supervised real-world image denoising.

### 3.1 Grid Self-Attention and Directional Self-Attention

For a set of tokens stored as $\boldsymbol{Z} = [\boldsymbol{z}_1; \cdots ; \boldsymbol{z}_L] \in \mathbb{R}^{L \times D}$, SA seeks to derive new token representations by assessing interdependence among every pair of input tokens. Consider SA in a multi-head setting [63]. For the $h$-th of $H$ attention head, all tokens undergo a linear transform resulting in $\boldsymbol{Q}_h, \boldsymbol{K}_h, \boldsymbol{V}_h \in \mathbb{R}^{L \times D}$, which represent queries, keys and values respectively:

$$(\boldsymbol{Q}_h, \boldsymbol{K}_h, \boldsymbol{V}_h) = (\boldsymbol{Z}\boldsymbol{W}_h^{\mathrm{Q}}, \boldsymbol{Z}\boldsymbol{W}_h^{\mathrm{K}}, \boldsymbol{Z}\boldsymbol{W}_h^{\mathrm{V}}), \tag{1}$$

where $\boldsymbol{W}_h^{\mathrm{Q}}, \boldsymbol{W}_h^{\mathrm{K}}, \boldsymbol{W}_h^{\mathrm{V}}$ are learnable matrices. Subsequently, attention weights are derived, determining the extent to which each token interacts with its counterparts. This is achieved through the calculation of similarity scores between queries and keys, leading to:

$$\mathrm{Head}_h = \mathrm{softmax}(\boldsymbol{Q}_h\boldsymbol{K}_h^T/\sqrt{D})\boldsymbol{V}_h. \tag{2}$$

To consolidate results from all attention heads, the output is given by

$$\mathrm{SA}(\boldsymbol{Z}) = \mathrm{concat}([\mathrm{Head}_1, \mathrm{Head}_2, \cdots, \mathrm{Head}_H])\boldsymbol{W}^O, \tag{3}$$

where $\boldsymbol{W}^O$ is a learnable matrix dedicated to fusing the results of the different attention heads.

SA involves pairwise comparison of tokens, which is costly when directly working on all spatial tokens from a feature tensor. As an acceleration scheme, grid SA [41] grids the tensor of shape $(H, W, C)$ into the shape $(G^2, \frac{HW}{G^2}, \mathrm{C})$ using a fixed $G \times G$ uniform grid, resulting in windows with adaptive size $\frac{H}{G} \times \frac{W}{G}$. Then SA is performed on the decomposed grid axis (*i.e.*, $G \times G$), corresponding to dilated, global spatial mixing of tokens. See Figure 2 (a) for an illustration. Note that grid SA does not satisfy the blind-spot property.

Our DSA also uses the gridding trick for acceleration; see Figure 2(b) for the illustration. Instead of using a half-plane attention window (Figure1(b)), DSA employs a half-plane grid to reduce the number of compared tokens for $\boldsymbol{p}$. Only a subset of equally spaced grids on the half-plane is involved, which lowers computational cost. To enhance diversity, adjacent tokens have different attention grids. For example, tokens of different colors correspond to different grids (Figure2 (b)). Each attention head shifts the grid differently, ensuring each token visits all grids in one pass.

Our DSA can effectively simulate different attention grids by rotating and shifting the input. Despite different definition of attention grid,, DSA and grid SA in our approach share the same parameters $\boldsymbol{W}_h^{\mathrm{Q}}, \boldsymbol{W}_h^{\mathrm{K}}, \boldsymbol{W}_h^{\mathrm{V}}, \boldsymbol{W}_{hh}$, allowing the introduction of the Siamese structure into our SelfFormer.

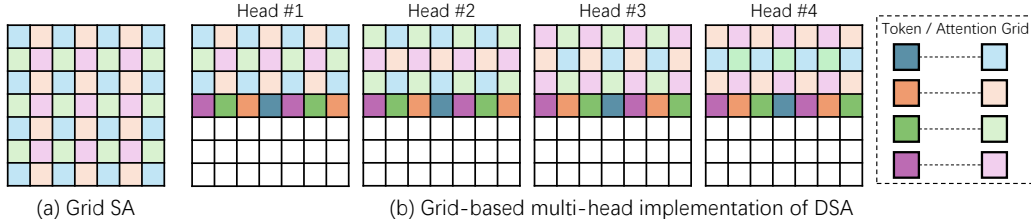

Figure 2: Illustration of grid SA and grid-based DSA. In grid SA, attention grids are scattered in all directions, with attention among tokens on same-colored grids. In DSA, attention grids are constrained to a half-plane, varying by token and attention head.

## 3.2 Architecture of SelfFormer

As illustrated in Figure 3, the proposed SelfFormer has a pseudo Siamese architecture consisting of two sub-NNs: SelfFormer-D (D for Directional) and SelfFormer-F (F for Full), both aiming at mapping an input noisy image to its clean counterpart. The SelfFormer-D consists of four branches, each performing DSA with one specific direction. The SelfFormer-F has only one branch which has the same structure as the branches in SelfFormer-D, except that its performs grid SA instead of DSA.

Each branch consists of six attention blocks (ABs) for performing DSA or grid SA, with two $1 \times 1$ convolutions at the beginning and end. The ABs are arranged in a U shape, with downsampling in the first half and upsampling in the second half. All branches in SelfFormer-D and SelfFormer-F share weights, except for the last $1 \times 1$ convolution. The last $1 \times 1$ convolution in SelfFormer-D integrates outputs from its four branches, resulting in a different input channel dimension from SelfFormer-F. Consequently, SelfFormer-D and SelfFormer-F have different structures but share identical learnable weights, except for the last layer, forming a pseudo-Siamese pair.

The pseudo-Siamese architecture benefits both training and testing. While SelfFormer-D's blind-spot mechanism aids self-supervised training to avoid overfitting, its restricted attention grid may weaken the utilization effectiveness of long-range dependency. In contrast, SelfFormer-F uses a full attention grid, avoiding this issue though losing the function of a BSN. Mutual learning between these sub-networks mitigates SelfFormer-D's structural bias, enhancing effectiveness. Additionally, SelfFormer-F utilizes a single path, as opposed to the four-path structure of SelfFormer-D, resulting in significantly faster performance. Due to these advantages, SelfFormer-F is employed for inference.

**Downsampling and upsampling:** The pairs of downsampling and upsampling used in SelfFormer can extend the receptive field in all directions. To maintain a blind-spot structure, we implement the same idea in [23] which attaches offsets to the downsampling layers. For a $2 \times 2$ average downsampling layer, we restrict the receptive field to extend upwards only by padding the input tensor with one row of zeros at top and cropping out the bottom row before operating downsampling.

**Attention blocks:** Each AB sequentially comprises a CA, a Feed-Forward Network (FFN), a DSA module, and another FFN. ABs involve multiple $1 \times 1$ convolutions and summation operations, which do not need specific treatment for blind-spot mechanism, as they neither change the receptive field nor spread information spatially. The CA is implemented as the NAFBlock [11] in SelfFormer-F, and we construct its blind-spot counterpart (called BSCA) for SelfFormer-D.

**Channel attention w/ and w/o blind-spot:** Channel attention re-calibrates global feature responses for primitive inputs by explicitly modeling inter-dependencies of channels, complementing the function of SA. We use NAFBlock [11] for channel attention, consisting of SimpleGate and Simplified Channel Attention (SCA). See Figure 3 for details. Given 2 feature tensors, $X_1$ and $X_2$, generated by two $3 \times 3$ convolutions on an input feature, SimpleGate outputs $X = X_1 \odot X_2$, where $\odot$ denotes element-wise multiplication, maintaining the receptive field. The $3 \times 3$ convolutions in SelfFormer-D are directional convolution [23] to own the blind-spot property. Then, SCA re-calibrates channels by multiplying each channel of $X$ with a weight scalar, calculated using a channel-wise Global Average Pooling (GAP) and a $1 \times 1$ convolution. As GAP squeezes all spatial values into a scalar and scale-multiplication do not spread spatial information, the SCA preserves the blind-spot property.

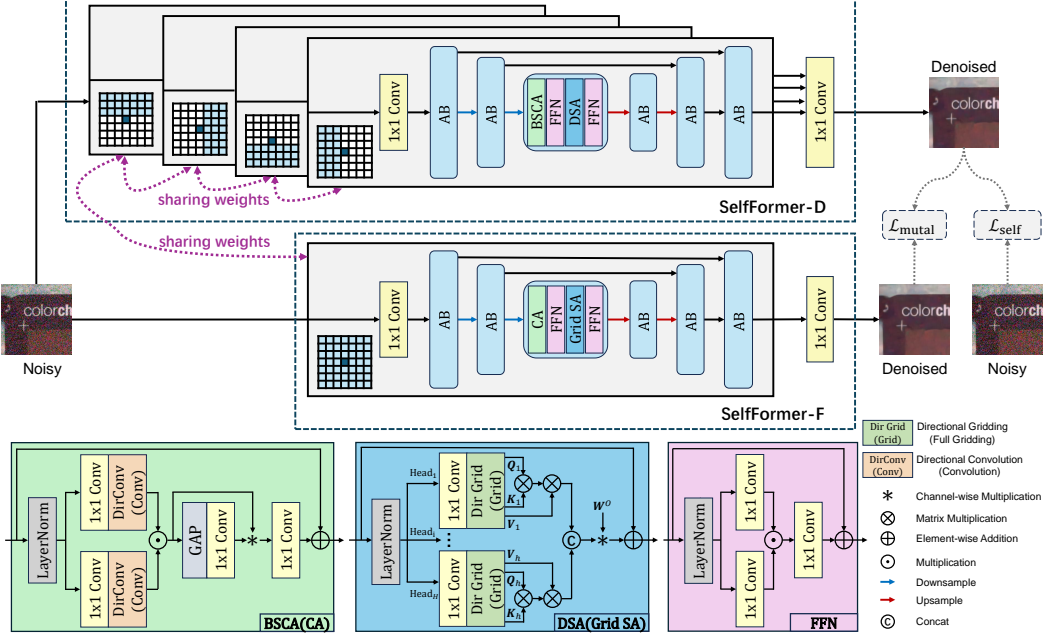

Figure 3: Architecture of the proposed SelfFormer.

## 3.3 Loss Function

The overall loss function consists of a self-reconstruction loss $\mathcal{L}_{\text{self}}$ and a mutual learning loss $\mathcal{L}_{\text{mutual}}$:

$$\mathcal{L}_{\text{total}} = \mathcal{L}_{\text{self}} + \lambda \mathcal{L}_{\text{mutual}}, \tag{4}$$

where $\lambda$ is set to 1. Let $\boldsymbol{x}$ denote a noisy image. Let $\mathcal{M}_{\text{D}}, \mathcal{M}_{\text{F}}$ denote the models of SelfFormer-D and SelfFormer-F, respectively. The self-reconstruction loss is then defined by

$$\mathcal{L}_{\text{self}} = \mathbb{E}_{\boldsymbol{x}} ||\mathcal{M}_{\text{D}}(\boldsymbol{x}) - \boldsymbol{x}||_1. \tag{5}$$

Due to the blind-sport structure of SelfFormer-D, $\mathcal{M}_{\text{D}}$ trained with this loss will not converge to the trivial identity mapping, but rather an effective denoiser. The mutual loss is defined by

$$\mathcal{L}_{\text{mutual}} = \mathbb{E}_{\boldsymbol{x}} ||\mathcal{M}_{\text{D}}(\boldsymbol{x}) - \mathcal{M}_{\text{F}}(\boldsymbol{x})||_1. \tag{6}$$

This loss enables the regularization effect of the two sub-networks to each other, leading to better generalization performance.

## 4 Experiments

### 4.1 Experimental setting

**Datasets:** Three widely-used real-world datasets are used for evaluation: SIDD [13], DND [14], and NIND [64]. SIDD is created by photographing a scene many times and using its mean as the GT clean image. The images of SIDD are captured by five different smartphone, and they are divided into non-overlapping subsets for training, validation, and test, respectively. As most works do, these samples are cropped into 24542 pairs of patches. the SIDD-Medium subset is chosen as training data, consisting of 320 noisy/clean image pairs. The validation subset, denoted by SIDD-Validation, consists of 1280 paired samples for hyper-parameter tuning and ablation study. The subset for test is denoted by SIDD-Benchmark, consisting of 1280 noisy samples. DND consists of 50 noisy-clean pairs, formed by shooting the same scene twice with different ISO values. The high-ISO images serve as noisy inputs and the corresponding low-ISO images serve as GT images. DND is used only for testing for performance evaluation. NIND is a real-world dataset consisting of clean-noisy image pairs captured at ISO levels of 3200, 4000, 5000, and 6400, with 22, 14, 13, and 79 pairs, respectively. For evaluation, we select ISO 3200 and ISO 5000, following the provided training/test split. Note

that as the GTs of SIDD-Benchmark and DND are not accessible to users. All denoising results are uploaded to the official websites of these two datasets for calculating the peak signal-to-noise ratio (PSNR) and the structural similarity (SSIM) index,

**Implementation details:** Our work is implemented on PyTorch1.10 and CUDA 11.8, which will be released upon paper acceptance. All experiments are conducted on an NVIDIA A6000 GPU. The grid size of SelfFormer-D is set to image size divided by $8$, and it doubles for SelfFormer-F. To better address noise correlation, we mask out the $4 \times 4$ half-plane neighboring locations around the center pixel within the attention window of the DSA of SelfFormer-D during training. This ensures that the pixels used in the attention window are at a distance from the central pixel, thereby reducing the noise correlation as the distance between the pixels increases. SelfFormer-D is optimized using Adam with a learning rate of $0.0001$, and that of SelfFormer-F is doubled. Other parameters of Adam are set to default. The entire model is trained for 30 epochs for full convergence.

## 4.2 Performance Evaluation on Real-world Denoising

We include an extensive list of image denoisers for comparison: *(a) Two representative non-learning-based methods*: BM3D [65] and WNNM [66]; *(b) Three classic supervised denoisers trained on synthetic noisy data*: DnCNN [1], CBDNet [12] and Zhou *et al.*[38]; *(c) Five supervised denoisers trained on real-world noisy images of SIDD-Medium*: DnCNN [1], VDN [7], AINDNet(R) [9], DANet [8] and NAFNet [11]; *(d) Three unpaired learning methods* GCBD [15], C2N+DIDN [16] and D-BSN+MWCNN [26]; *(e) Four zero-shot denoisers*: Self2Self [59], NoisyAsClean [60], ScoreDVI [62] and MASH [67]; and *(f) Twelve Self-supervised denoisers*: Noise2Void [21], Noise2Self [22], Laine *et al.* [23], Recorrupted2Recorrupted (R2R) [45], CVF-SID [53], AP-BSN [27], LG-BPN [28], MM-BSN [40], SASL [39], SS-BSN [29], C-BSN [68], and PUCA [30].

Table 1 compares the quantitative results of different methods, where we mark in bold the best results among all methods that do not call any clean images for training, including the non-learning-based, zero-shot and self-supervised methods. It can be seen from Table 1 that our SelfFormer achieved the best results on all the benchmark datasets, in terms of both PSNR and SSIM.

R2R and input-masking-based BSNs, including Noise2Void, Noise2Self, and Laine *et al.*'s, rely heavily on the spatial independence of noise, resulting in poor performance on real-world images with locally-correlated noise. In contrast, SelfFormer leverages interactions of distant pixels via its transformer architecture during training, leading to significant performance gains. Compared to AP-BSN, MM-BSN, SASL, and PUCA, SelfFormer's superior performance comes from utilizing the transformer's long-range perception capability for relating distant pixels.

Regarding SA-based BSNs, including LG-BSN and SS-BSN, our SelfFormer performs noticeably better than LG-BSN. Although SS-BSN matches SelfFormer in SSIM on SIDD, its PSNR performance on DND is noticeably worse. These results confirm the higher effectiveness of our SSA module compared to the dilated SA in LG-BSN and the plain grid SA in SS-BSN.

**Visual comparison:** Refer to Figure 4 for a visualization of denoising results from top BSN-based methods. We selecte the images with richly textured regions from SIDD-benchmark and DND for comparison. Laine *et al.*failed to break spatial noise correlation, resulting in undesirable denoising. While achieving adequate global denoising, AP-BSN and LG-BPN generate artifacts, and PUCA smooths out some details. We successfully recovers the detailed texture of the clean image by proposed DSA. Unlike SIDD-Benchmark and DND, the SIDD-Validation provides GT images and we choose some images of it for evaluation. See Figure 5 for a visual comparison of these images relies on global information. Both AP-BSN and LG-BPN fail to separate spatial noise from image details, leading to deficient denoising. In contrast, SelfFormer-F effectively eliminate spatial correlation noise while preventing the creation of erroneous flat regions.

**Computational complexity:** Table 2 compares the computational complexity of transformer-based self-supervised denoisers and top CNN-based denoiser, in terms of model size (number of parameters) and inference time for a $256 \times 256$ image. SelfFormer has a smaller size than the second-best performer, PUCA, and is much faster than transformer-based methods: LG-BPN and SS-BSN.

Table 1: PSNR(dB)/SSIM results on SIDD benchmark, DND benchmark and NIND dataset. We report the official results from the benchmarks' websites whenever possible. The results of Zhou *et al.* are quoted from [27]. The results of Noise2Void and Noise2Self are quoted from [45]. The results of DnCNN and Laine *et al.* are obtained by code re-running. **Bold** denotes the best results in **clean-image-free** methods.

| | Type | Method | SIDD | DND | NIND ISO3200 | NIND ISO5000 |
|---|---|---|---|---|---|---|
| Requiring clean images | Supervised (Synthetic pairs) | DnCNN [1] | 23.66/.583 | 32.43/.790 | - | - |
| | | CBDNet [12] | 33.28/.868 | 38.05/.942 | - | - |
| | | Zhou *et al.* [38] | 34.00/.898 | 38.40/.945 | - | - |
| | Supervised (Real pairs) | DnCNN [1] | 35.87/.842 | 35.74/.903 | - | - |
| | | VDN [7] | 39.26/.955 | 39.38/.952 | - | - |
| | | AINDNet(R) [9] | 38.84/.951 | 39.34/.952 | - | - |
| | | DANet [8] | 39.43/.956 | 39.58/.955 | - | - |
| | | NAFNet [11] | 40.30/.961 | - | - | - |
| | Unpaired | GCBD [15] | - | 35.58/.922 | - | - |
| | | C2N+DIDN [16] | 35.35/.937 | 37.28/.924 | - | - |
| | | D-BSN+MWCNN [26] | - | 37.93/.937 | - | - |
| Free from clean images | Non-learning | BM3D [65] | 25.65/.685 | 34.51/.851 | - | - |
| | | WNNM [66] | 25.78/.809 | 34.67/.865 | - | - |
| | Zero-shot | Self2Self [59] | 29.51/.651 | - | - | - |
| | | NAC [60] | - | 36.20/.925 | - | - |
| | | ScoreDVI [62] | 34.60/.920 | - | - | - |
| | | MASH [67] | 34.78/.900 | - | - | - |
| | Self-supervised | Noise2Void [21] | 27.68/.668 | - | 28.42/.766 | 27.04/.658 |
| | | Noise2Self [22] | 29.56/.808 | - | - | - |
| | | Laine *et al.* [23] | 30.14/.823 | 35.13/.862 | - | - |
| | | R2R [45] | 34.78/.898 | - | - | - |
| | | CVF-SID [53] | 34.71/.917 | 36.50/.924 | - | - |
| | | AP-BSN [27] | 36.91/.931 | 38.09/.937 | 34.41/.854 | 33.49/**.847** |
| | | LG-BPN [28] | 37.28/.936 | 38.43/.942 | 33.94/.840 | 33.33/.831 |
| | | MM-BSN [40] | 37.37/.936 | 38.46/.940 | - | - |
| | | SASL [39] | 37.41/.934 | 38.18/.938 | - | - |
| | | SS-BSN [29] | 37.42/**.937** | 38.46/.940 | - | - |
| | | C-BSN [68] | 37.43/.936 | 38.62/.942 | 34.33/.855 | 33.52/.839 |
| | | PUCA [30] | 37.54/.936 | 38.83/.942 | 34.24/.854 | 33.49/.840 |
| | | SelfFormer [Ours] | **37.69/.937** | **38.92/.943** | **34.43/.857** | **33.55/.847** |

Table 2: Computational complexity comparison in terms of model size and inference time.

| | CNN-based Method | | | Transformer-based Method | | |
|---|---|---|---|---|---|---|
| | AP-BSN | MM-BSN | PUCA | LG-BPN | SS-BSN | SelfFormer-F |
| PSNR(dB) | 36.87 | 37.38 | 37.51 | 37.28 | 37.32 | 37.63 |
| #Param(M) | 3.7 | 5.7 | 12.8 | 4.8 | 6.4 | 10.8 |
| Time(ms) | 382 | 539 | 529 | 5208 | 3976 | 1812 |

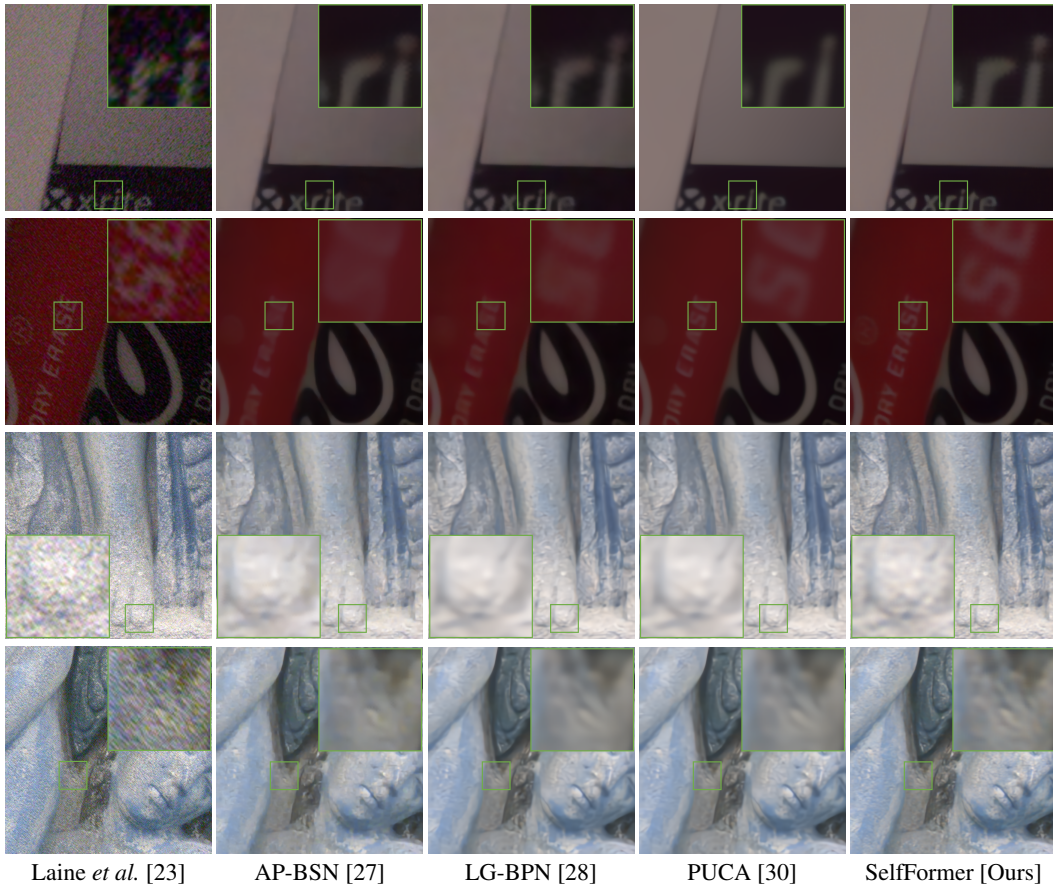

| Laine *et al.* [23] | AP-BSN [27] | LG-BPN [28] | PUCA [30] | SelfFormer [Ours] |

Figure 4: Visual inspection of the results of some samples from SIDD-Benchmark and DND.

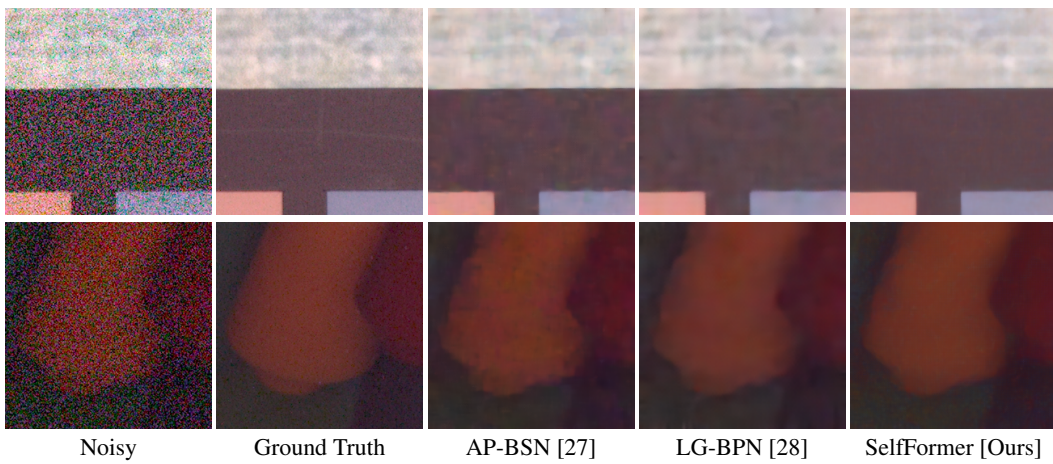

| Noisy | Ground Truth | AP-BSN [27] | LG-BPN [28] | SelfFormer [Ours] |

Figure 5: Visual comparison on samples from SIDD-Validation.

## 4.3 Ablation Studies

Following existing works, we conduct the ablation study on SIDD-Validation. We consider four baselines, summarized in Table 3. (a) DSA→Grid SA: The effectiveness of DSA is tested by replacing it with grid SA in SelfFormer-D and reducing the branch number to one, resulting in a true Siamese pair structure similar to SelfFormer-S. Despite some performance gain from mutual learning, this baseline performs significantly worse than the original model. This is due to the arbitrary feature flow, which loses the blind-spot property. (b) w/o CA: We replace all BSCA (CA) modules with DSA (Grid SA) modules. (c) Only SelfFormer-D: We train SelfFormer-D without $\mathcal{L}_{\mathrm{mutual}}$ and test with it. Without the Siamese structure and mutual learning, the performance noticeably decreases due to the limitations of the constrained attention grids in SelfFormer and the loss of regularization from mutual learning. (d) Only SelfFormer-F: We train SelfFormer-F only using $\mathcal{L}_{\mathrm{self}}$ and test with it. Lacking any blind-spot mechanism and regularization from mutual learning, this network performs very poorly, indicating overfitting to noisy image. (e) w/o DSA: All DSA (Grid SA) modules are replaced with the BSCA (CA) modules. Since the blind spot property is preserved, the performance remains better than "Only SelfFormer-F", which lacks this property. (f) w/o WeightShare: In this case, the SelfFormer-D and SelfFormer-F do not share weight, leading to a performance loss. This suggests that the weight sharing not only reduces the the number of parameters but also provides a regularization effect.

Table 3: PSNR(dB)/SSIM results of ablation studies on SIDD-Validation.

| DSA→Grid SA | w/o CA | Only SelfFormer-D | Only SelfFormer-F | w/o DSA | w/o WeightShare | Original |
|---|---|---|---|---|---|---|
| 23.96/0.336 | 37.54/0.881 | 37.29/0.879 | 23.66/0.328 | 37.22/0.880 | 37.45/0.880 | **37.63/0.882** |

Table 4 shows the experimental results on SIDD-Validation, with varying grid sizes and loss function weight $\lambda$. The results indicate that increasing the grid size can improve performance, as more pixels are utilized in DSA and grid SA. However, setting the loss weight $\lambda$ too small or too large results in a noticeable decrease in performance.

Table 4: Results with varying grid sizes and loss function weights on SIDD-Validation

| Grid Size | | | Loss Weight | | |
|---|---|---|---|---|---|
| Grid size | PSNR(dB) | SSIM | Loss weight | PSNR(dB) | SSIM |
| 8 | 37.43 | 0.874 | 0.01 | 37.35 | 0.879 |
| 12 | 37.51 | 0.879 | 0.1 | 37.57 | 0.881 |
| 16 | 37.63 | 0.882 | 1 | 37.63 | 0.882 |
| 20 | 37.65 | 0.882 | 10 | 37.51 | 0.880 |

## 5 Conclusion

We presented a transformer-based self-supervised framework for real-world image denoising, optimizing its performance by exploiting distant pixel interactions in transformer to reduce noise correlation. Our key innovation is a DSA module using a half-plane grid for SA, creating a blind-spot structure. A Siamese architecture with mutual learning addressed the performance impact caused by the restricted attention grid in DSA. Experiments on benchmark datasets show our method outperforms existing self-supervised and clean-image-free methods.

While SelfFormer outperforms all related self-supervised denoisers, transformer-based models generally incur higher computational costs compared to CNN-based models. Additionally, there remains a significant performance gap between SelfFormer and supervised denoisers. In the future, we aim to improve both the performance and efficiency of transformer-based denoiser.

## Acknowledgments

Yuhui Quan would like to acknowledge the support from National Natural Science Foundation of China under Grant 62372186, Natural Science Foundation of Guangdong Province under Grants 2022A1515011755 and 2023A1515012841, Fundamental Research Funds for the Central Universities under Grant x2jsD2230220, and National Key Research and Development Program of China under 2024YFE0105400. Hui Ji would like to acknowledge the support from Singapore MOE Academic Research Fund (AcRF) Tier 1 Research Grant A-8000981-00-00.

## Footnotes

\*email: csyhquan@scut.edu.cn (Y. Quan); cszhengtx@mail.scut.edu.cn (T. Zheng); matjh@nus.edu.sg (H. Ji).

## References

[1] Kai Zhang, Wangmeng Zuo, Yunjin Chen, Deyu Meng, and Lei Zhang. Beyond a gaussian denoiser: Residual learning of deep cnn for image denoising. *IEEE Transactions on Image Processing*, 26(7):3142–3155, 2017.

[2] Xiaolong Wang, Ross Girshick, Abhinav Gupta, and Kaiming He. Non-local neural networks. In *Proceedings of the IEEE/CVF Conference on Computer Vision and Pattern Recognition*, pages 7794–7803, 2018.

[3] Pengju Liu, Hongzhi Zhang, Kai Zhang, Liang Lin, and Wangmeng Zuo. Multi-level wavelet-cnn for image restoration. In *Proceedings of the IEEE/CVF Conference on Computer Vision and Pattern Recognition Workshops*, pages 773–782, 2018.

[4] Kai Zhang, Wangmeng Zuo, and Lei Zhang. Ffdnet: Toward a fast and flexible solution for cnn-based image denoising. *IEEE Transactions on Image Processing*, 27(9):4608–4622, 2018.

[5] Shuhang Gu, Yawei Li, Luc Van Gool, and Radu Timofte. Self-guided network for fast image denoising. In *Proceedings of the IEEE/CVF International Conference on Computer Vision*, pages 2511–2520, 2019.

[6] Saeed Anwar and Nick Barnes. Real image denoising with feature attention. In *Proceedings of the IEEE/CVF International Conference on Computer Vision*, pages 3155–3164, 2019.

[7] Zongsheng Yue, Hongwei Yong, Qian Zhao, Deyu Meng, and Lei Zhang. Variational denoising network: Toward blind noise modeling and removal. *Advances in Neural Information Processing Systems*, 32, 2019.

[8] Zongsheng Yue, Qian Zhao, Lei Zhang, and Deyu Meng. Dual adversarial network: Toward real-world noise removal and noise generation. In *Proceedings of the European Conference on Computer Vision*, pages 41–58. Springer, 2020.

[9] Yoonsik Kim, Jae Woong Soh, Gu Yong Park, and Nam Ik Cho. Transfer learning from synthetic to real-noise denoising with adaptive instance normalization. In *Proceedings of the IEEE/CVF Conference on Computer Vision and Pattern Recognition*, pages 3482–3492, 2020.

[10] Syed Waqas Zamir, Aditya Arora, Salman Khan, Munawar Hayat, Fahad Shahbaz Khan, Ming-Hsuan Yang, and Ling Shao. Multi-stage progressive image restoration. In *Proceedings of the IEEE/CVF Conference on Computer Vision and Pattern Recognition*, pages 14821–14831, 2021.

[11] Liangyu Chen, Xiaojie Chu, Xiangyu Zhang, and Jian Sun. Simple baselines for image restoration. In *Proceedings of the European Conference on Computer Vision*, pages 17–33. Springer, 2022.

[12] Shi Guo, Zifei Yan, Kai Zhang, Wangmeng Zuo, and Lei Zhang. Toward convolutional blind denoising of real photographs. In *Proceedings of the IEEE/CVF Conference on Computer Vision and Pattern Recognition*, pages 1712–1722, 2019.

[13] Abdelrahman Abdelhamed, Stephen Lin, and Michael S Brown. A high-quality denoising dataset for smartphone cameras. In *Proceedings of the IEEE/CVF Conference on Computer Vision and Pattern Recognition*, pages 1692–1700, 2018.

[14] Tobias Plotz and Stefan Roth. Benchmarking denoising algorithms with real photographs. In *Proceedings of the IEEE/CVF Conference on Computer Vision and Pattern Recognition*, pages 1586–1595, 2017.

[15] Jingwen Chen, Jiawei Chen, Hongyang Chao, and Ming Yang. Image blind denoising with generative adversarial network based noise modeling. In *Proceedings of the IEEE/CVF Conference on Computer Vision and Pattern Recognition*, pages 3155–3164, 2018.

[16] Geonwoon Jang, Wooseok Lee, Sanghyun Son, and Kyoung Mu Lee. C2N: Practical generative noise modeling for real-world denoising. In *Proceedings of the IEEE/CVF International Conference on Computer Vision*, pages 2350–2359, 2021.

[17] Abdelrahman Abdelhamed, Marcus A Brubaker, and Michael S Brown. Noise flow: Noise modeling with conditional normalizing flows. In *Proceedings of the IEEE/CVF International Conference on Computer Vision*, pages 3165–3173, 2019.

[18] Jaakko Lehtinen, Jacob Munkberg, Jon Hasselgren, Samuli Laine, Tero Karras, Miika Aittala, and Timo Aila. Noise2noise: Learning image restoration without clean data. In *Proceedings of the International Conference on Machine Learning*, pages 2965–2974. PMLR, 2018.

[19] Magauiya Zhussip, Shakarim Soltanayev, and Se Young Chun. Extending stein's unbiased risk estimator to train deep denoisers with correlated pairs of noisy images. *Advances in Neural Information Processing Systems*, 32, 2019.

[20] Ali Maleky, Shayan Kousha, Michael S Brown, and Marcus A Brubaker. Noise2noiseflow: Realistic camera noise modeling without clean images. In *Proceedings of the IEEE/CVF Conference on Computer Vision and Pattern Recognition*, pages 17632–17641, 2022.

[21] Alexander Krull, Tim-Oliver Buchholz, and Florian Jug. Noise2void-learning denoising from single noisy images. In *Proceedings of the IEEE/CVF Conference on Computer Vision and Pattern Recognition*, pages 2129–2137, 2019.

[22] Joshua Batson and Loic Royer. Noise2self: Blind denoising by self-supervision. In *Proceedings of the International Conference on Machine Learning*, pages 524–533. PMLR, 2019.

[23] Samuli Laine, Tero Karras, Jaakko Lehtinen, and Timo Aila. High-quality self-supervised deep image denoising. *Advances in Neural Information Processing Systems*, 32, 2019.

[24] Yaochen Xie, Zhengyang Wang, and Shuiwang Ji. Noise2same: Optimizing a self-supervised bound for image denoising. *Advances in Neural Information Processing Systems*, 33:20320–20330, 2020.

[25] Zejin Wang, Jiazheng Liu, Guoqing Li, and Hua Han. Blind2unblind: Self-supervised image denoising with visible blind spots. In *Proceedings of the IEEE/CVF Conference on Computer Vision and Pattern Recognition*, pages 2027–2036, 2022.

[26] Xiaohe Wu, Ming Liu, Yue Cao, Dongwei Ren, and Wangmeng Zuo. Unpaired learning of deep image denoising. In *Proceedings of the European Conference on Computer Vision*, pages 352–368. Springer, 2020.

[27] Wooseok Lee, Sanghyun Son, and Kyoung Mu Lee. Ap-BSN: Self-supervised denoising for real-world images via asymmetric pd and blind-spot network. In *Proceedings of the IEEE/CVF Conference on Computer Vision and Pattern Recognition*, pages 17725–17734, 2022.

[28] Zichun Wang, Ying Fu, Ji Liu, and Yulun Zhang. LG-BPN: Local and global blind-patch network for self-supervised real-world denoising. In *Proceedings of the IEEE/CVF Conference on Computer Vision and Pattern Recognition*, pages 18156–18165, 2023.

[29] Young-Joo Han and Ha-Jin Yu. SS-BSN: Attentive blind-spot network for self-supervised denoising with nonlocal self-similarity. In *Proceedings of the International Joint Conference on Artificial Intelligence*, pages 800–809, 2023.

[30] Hyemi Jang, Junsung Park, Dahuin Jung, Jaihyun Lew, Ho Bae, and Sungroh Yoon. Puca: Patch-unshuffle and channel attention for enhanced self-supervised image denoising. *Advances in Neural Information Processing Systems*, 36, 2024.

[31] Ze Liu, Yutong Lin, Yue Cao, Han Hu, Yixuan Wei, Zheng Zhang, Stephen Lin, and Baining Guo. Swin transformer: Hierarchical vision transformer using shifted windows. In *Proceedings of the IEEE/CVF International Conference on Computer Vision*, pages 10012–10022, 2021.

[32] Zheng Chen, Yulun Zhang, Jinjin Gu, Linghe Kong, Xin Yuan, et al. Cross aggregation transformer for image restoration. *Advances in Neural Information Processing Systems*, 35:25478–25490, 2022.

[33] Zhendong Wang, Xiaodong Cun, Jianmin Bao, Wengang Zhou, Jianzhuang Liu, and Houqiang Li. Uformer: A general u-shaped transformer for image restoration. In *Proceedings of the IEEE/CVF Conference on Computer Vision and Pattern Recognition*, pages 17683–17693, 2022.

[34] Syed Waqas Zamir, Aditya Arora, Salman Khan, Munawar Hayat, Fahad Shahbaz Khan, and Ming-Hsuan Yang. Restormer: Efficient transformer for high-resolution image restoration. In *Proceedings of the IEEE/CVF Conference on Computer Vision and Pattern Recognition*, pages 5728–5739, 2022.

[35] Haoyu Chen, Jinjin Gu, Yihao Liu, Salma Abdel Magid, Chao Dong, Qiong Wang, Hanspeter Pfister, and Lei Zhu. Masked image training for generalizable deep image denoising. In *Proceedings of the IEEE/CVF Conference on Computer Vision and Pattern Recognition*, pages 1692–1703, 2023.

[36] Mikhail Papkov and Pavel Chizhov. SwinIA: Self-supervised blind-spot image denoising with zero convolutions. *arXiv preprint arXiv:2305.05651*, 2023.

[37] Zhengzhong Tu, Hossein Talebi, Han Zhang, Feng Yang, Peyman Milanfar, Alan Bovik, and Yinxiao Li. Maxim: Multi-axis mlp for image processing. In *Proceedings of the IEEE/CVF Conference on Computer Vision and Pattern Recognition*, pages 5769–5780, 2022.

[38] Yuqian Zhou, Jianbo Jiao, Haibin Huang, Yang Wang, Jue Wang, Honghui Shi, and Thomas Huang. When AWGN-based denoiser meets real noises. In *Proceedings of the AAAI Conference on Artificial Intelligence*, volume 34, pages 13074–13081, 2020.

[39] Junyi Li, Zhilu Zhang, Xiaoyu Liu, Chaoyu Feng, Xiaotao Wang, Lei Lei, and Wangmeng Zuo. Spatially adaptive self-supervised learning for real-world image denoising. In *Proceedings of the IEEE/CVF Conference on Computer Vision and Pattern Recognition*, pages 9914–9924, 2023.

[40] Dan Zhang, Fangfang Zhou, Yuwen Jiang, and Zhengming Fu. Mm-bsn: Self-supervised image denoising for real-world with multi-mask based on blind-spot network. In *Proceedings of the IEEE/CVF Conference on Computer Vision and Pattern Recognition Workshops*, pages 4189–4198. IEEE, 2023.

[41] Zhengzhong Tu, Hossein Talebi, Han Zhang, Feng Yang, Peyman Milanfar, Alan Bovik, and Yinxiao Li. Maxvit: Multi-axis vision transformer. In *Proceedings of the European Conference on Computer Vision*, pages 459–479. Springer, 2022.

[42] Christopher A Metzler, Ali Mousavi, Reinhard Heckel, and Richard G Baraniuk. Unsupervised learning with stein's unbiased risk estimator. *arXiv preprint arXiv:1805.10531*, 2018.

[43] Thierry Blu and Florian Luisier. The sure-let approach to image denoising. *IEEE Transactions on Image Processing*, 16(11):2778–2786, 2007.

[44] Nick Moran, Dan Schmidt, Yu Zhong, and Patrick Coady. Noisier2noise: Learning to denoise from unpaired noisy data. In *Proceedings of the IEEE/CVF Conference on Computer Vision and Pattern Recognition*, pages 12064–12072, 2020.

[45] Tongyao Pang, Huan Zheng, Yuhui Quan, and Hui Ji. Recorrupted-to-recorrupted: unsupervised deep learning for image denoising. In *Proceedings of the IEEE/CVF Conference on Computer Vision and Pattern Recognition*, pages 2043–2052, 2021.

[46] Yi Zhang, Dasong Li, Ka Lung Law, Xiaogang Wang, Hongwei Qin, and Hongsheng Li. IDR: Self-supervised image denoising via iterative data refinement. In *Proceedings of the IEEE/CVF Conference on Computer Vision and Pattern Recognition*, pages 2098–2107, 2022.

[47] Huan Zheng, Tongyao Pang, and Hui Ji. Unsupervised deep video denoising with untrained network. In *Proceedings of the AAAI Conference on Artificial Intelligence*, volume 37, pages 3651–3659, 2023.

[48] Tao Huang, Songjiang Li, Xu Jia, Huchuan Lu, and Jianzhuang Liu. Neighbor2neighbor: Self-supervised denoising from single noisy images. In *Proceedings of the IEEE/CVF Conference on Computer Vision and Pattern Recognition*, pages 14781–14790, 2021.

[49] Kwanyoung Kim and Jong Chul Ye. Noise2score: tweedie's approach to self-supervised image denoising without clean images. *Advances in Neural Information Processing Systems*, 34:864–874, 2021.

[50] Kwanyoung Kim, Taesung Kwon, and Jong Chul Ye. Noise distribution adaptive self-supervised image denoising using tweedie distribution and score matching. In *Proceedings of the IEEE/CVF Conference on Computer Vision and Pattern Recognition*, pages 2008–2016, 2022.

[51] Yutong Xie, Mingze Yuan, Bin Dong, and Quanzheng Li. Unsupervised image denoising with score function. *Advances in Neural Information Processing Systems*, 36, 2024.

[52] Aapo Hyvärinen and Peter Dayan. Estimation of non-normalized statistical models by score matching. *Journal of Machine Learning Research*, 6(4), 2005.

[53] Reyhaneh Neshatavar, Mohsen Yavartanoo, Sanghyun Son, and Kyoung Mu Lee. Cvf-sid: Cyclic multi-variate function for self-supervised image denoising by disentangling noise from image. In *Proceedings of the IEEE/CVF Conference on Computer Vision and Pattern Recognition*, pages 17583–17591, 2022.

[54] Michael Elad and Michal Aharon. Image denoising via sparse and redundant representations over learned dictionaries. *IEEE Trans. Image Process.*, 15(12):3736–3745, 2006.

[55] Chenglong Bao, Jian-Feng Cai, and Hui Ji. Fast sparsity-based orthogonal dictionary learning for image restoration. In *Proc. ICCV*, pages 3384–3391, 2013.

[56] Chenglong Bao, Hui Ji, Yuhui Quan, and Zuowei Shen. Dictionary learning for sparse coding: Algorithms and convergence analysis. *Trans. Pattern Anal. Mach. Intell.*, 38(7):1356–1369, 2015.

[57] Jian-Feng Cai, Hui Ji, Zuowei Shen, and Gui-Bo Ye. Data-driven tight frame construction and image denoising. *Applied and Computational Harmonic Analysis*, 37(1):89–105, 2014.

[58] Dmitry Ulyanov, Andrea Vedaldi, and Victor Lempitsky. Deep image prior. In *Proceedings of the IEEE Conference on Computer Vision and Pattern Recognition*, pages 9446–9454, 2018.

[59] Yuhui Quan, Mingqin Chen, Tongyao Pang, and Hui Ji. Self2self with dropout: Learning self-supervised denoising from single image. In *Proceedings of the IEEE/CVF Conference on Computer Vision and Pattern Recognition*, pages 1890–1898, 2020.

[60] Jun Xu, Yuan Huang, Ming-Ming Cheng, Li Liu, Fan Zhu, Zhou Xu, and Ling Shao. Noisy-as-clean: Learning self-supervised denoising from corrupted image. *IEEE Transactions on Image Processing*, 29:9316–9329, 2020.

[61] Jason Lequyer, Reuben Philip, Amit Sharma, Wen-Hsin Hsu, and Laurence Pelletier. A fast blind zero-shot denoiser. *Nature Machine Intelligence*, 4(11):953–963, 2022.

[62] Jun Cheng, Tao Liu, and Shan Tan. Score priors guided deep variational inference for unsupervised real-world single image denoising. In *Proceedings of the IEEE/CVF International Conference on Computer Vision*, pages 12937–12948, 2023.

[63] Ashish Vaswani, Noam Shazeer, Niki Parmar, Jakob Uszkoreit, Llion Jones, Aidan N Gomez, Łukasz Kaiser, and Illia Polosukhin. Attention is all you need. *Advances in Neural Information Processing Systems*, 30, 2017.

[64] Benoit Brummer and Christophe De Vleeschouwer. Natural image noise dataset. In *Proceedings of the IEEE/CVF Conference on Computer Vision and Pattern Recognition Workshops*, pages 0–0, 2019.

[65] Kostadin Dabov, Alessandro Foi, Vladimir Katkovnik, and Karen Egiazarian. Image denoising by sparse 3-D transform-domain collaborative filtering. *IEEE Transactions on Image Processing*, 16(8):2080–2095, 2007.

[66] Shuhang Gu, Lei Zhang, Wangmeng Zuo, and Xiangchu Feng. Weighted nuclear norm minimization with application to image denoising. In *Proceedings of the IEEE/CVF Conference on Computer Vision and Pattern Recognition*, pages 2862–2869, 2014.

[67] Hamadi Chihaoui and Paolo Favaro. Masked and shuffled blind spot denoising for real-world images. In *Proceedings of the IEEE/CVF Conference on Computer Vision and Pattern Recognition*, pages 3025–3034, 2024.

[68] Linwei Fan, Jin Cui, Huiyu Li, Xiaoyu Yan, Hui Liu, and Caiming Zhang. Complementary blind-spot network for self-supervised real image denoising. *IEEE Transactions on Circuits and Systems for Video Technology*, 2024.

# A Appendix

## A.1 Broader Impacts

The proposed self-supervised learning method for denoising images has the potential to impact a wide range of applications, including surveillance security, scientific research, and digital media restoration. By improving the signal-to-noise ratio of images, our research can facilitate deeper insights and more effective interventions in these fields. In surveillance, higher clarity images can enhance public safety by providing more detailed visual information. However, this also raises concerns about privacy and the potential for mass surveillance. In scientific research, improved image quality can lead to better data and more significant discoveries, though there is a risk that overly processed images could misrepresent the original data. In digital media restoration, while the technique helps preserve cultural heritage, it also poses the risk of altering historical records. Despite these possible concerns, our goal is to contribute to enhancing image clarity in critical areas such as public safety, scientific research, and cultural preservation. We emphasize the responsible application and continuous improvement of this technology to mitigate potential risks and maximize its positive impact.

## A.2 Additional Results

Table A1: Quantitative results of self-supervised methods on SIDD-Validation.

| Metric | N2V | R2R | CVF-SID | AP-BSN | LG-BPN | SASL | PUCA | C-BSN | SelfFormer [Ours] |
|---|---|---|---|---|---|---|---|---|---|
| PSNR(dB) | 27.06 | 35.04 | 34.15 | 36.73 | 37.31 | 37.39 | 37.49 | 37.51 | 37.63 |
| SSIM | 0.651 | 0.844 | 0.871 | 0.878 | 0.884 | 0.875 | 0.880 | 0.885 | 0.882 |

Table A2: Computational complexity comparison of different methods in terms of training time, memory usage in training, and memory usage in inference.

| | CNN-based Methods | | | Transformer-based Methods | | |
|---|---|---|---|---|---|---|
| | AP-BSN | MM-BSN | PUCA | LG-BPN | SS-BSN | SelfFormer [Ours] |
| Training Time (h) | 5.086 | 11.291 | 10.227 | 6.313 | 49.483 | 56.266 |
| Training Mem. (G) | 0.574 | 1.349 | 2.549 | 1.884 | 0.873 | 9.161 |
| Inference Mem. (G) | 0.664 | 1.310 | 1.200 | 12.072 | 3.492 | 2.806 |

For further analysis, we conducted an additional quantitative comparison on the SIDD-Validation dataset, as shown in Table A1. Moreover, we provide a detailed complexity comparison in Table A2, demonstrating that our method's memory usage during inference is lower than that of the other two related transformer-based methods, particularly when compared to LG-BPN. However, due to the inclusion of SelfFormer-D, which incorporates multiple paths for DSA, the memory usage of our method during training remains relatively high. This confirms one of our key motivations for introducing the pseudo-Siamese architecture, namely to significantly reduce memory consumption from the training phase to inference. For additional visual comparisons on DND, SIDD Validation and SIDD-Benchmark, please refer to Fig A1, A2, and A3.

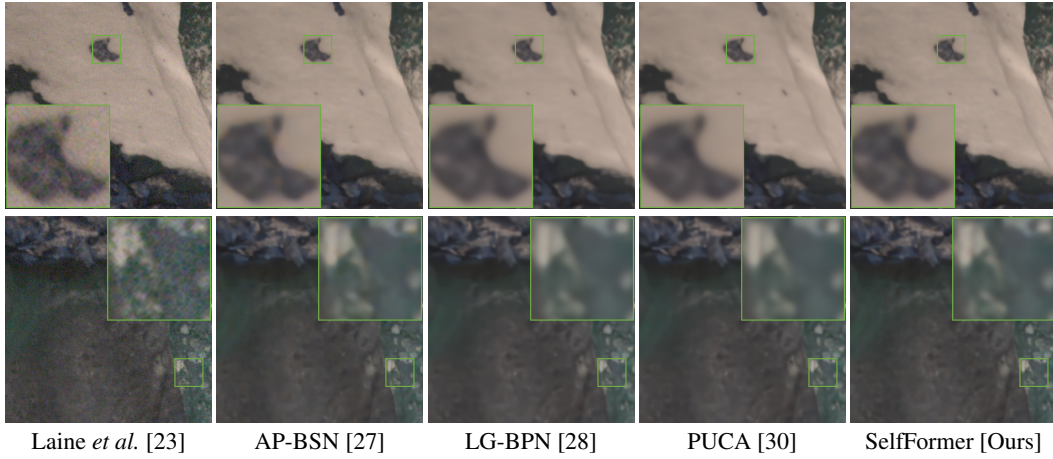

| Laine *et al.* [23] | AP-BSN [27] | LG-BPN [28] | PUCA [30] | SelfFormer [Ours] |

Figure A1: More qualitative comparison on samples from DND.

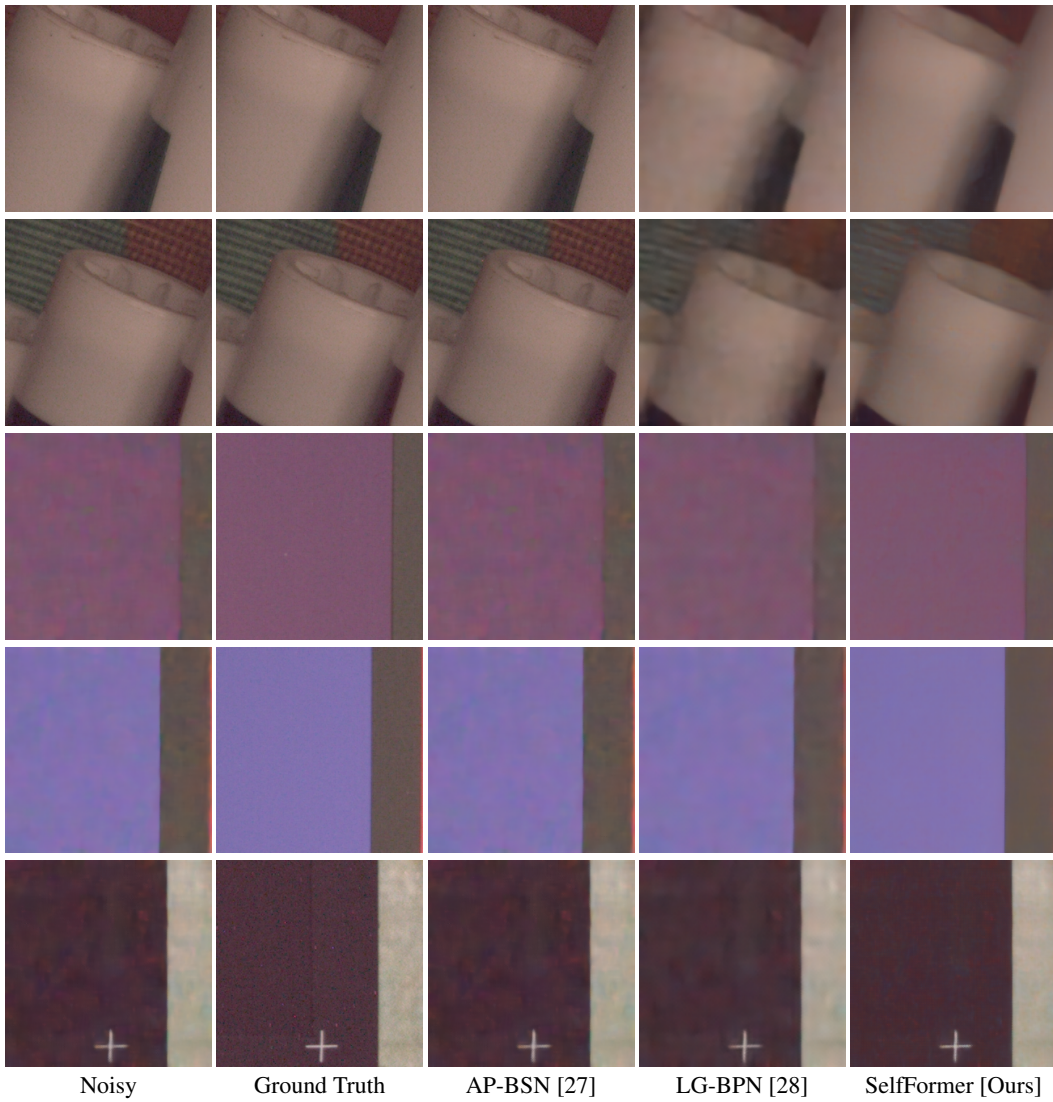

| Noisy | Ground Truth | AP-BSN [27] | LG-BPN [28] | SelfFormer [Ours] |

Figure A2: More qualitative comparison on samples from SIDD-Validation.

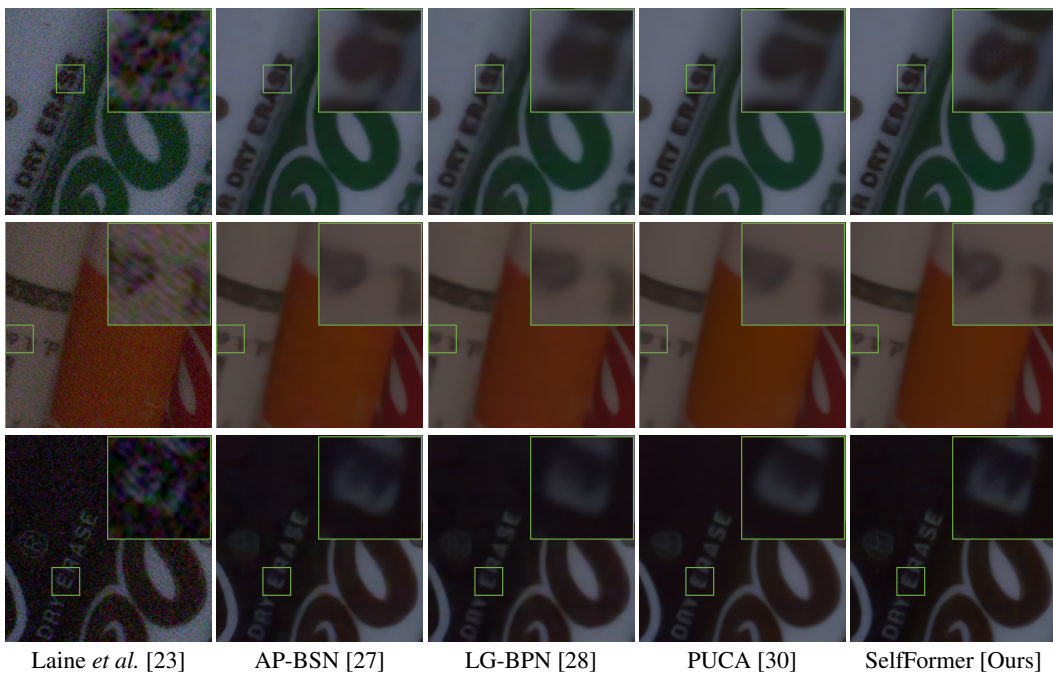

| Laine *et al.* [23] | AP-BSN [27] | LG-BPN [28] | PUCA [30] | SelfFormer [Ours] |

Figure A3: More qualitative comparison on samples from SIDD-Benchmark.

